# Unsupervised Classification of 3D Objects from 2D Views

**Satoshi Suzuki**    **Hiroshi Ando**
ATR Human Information Processing Research Laboratories
2-2 Hikaridai, Seika-cho, Soraku-gun, Kyoto 619-02, Japan
satoshi@hip.atr.co.jp, ando@hip.atr.co.jp

## Abstract

This paper presents an unsupervised learning scheme for categorizing 3D objects from their 2D projected images. The scheme exploits an auto-associative network's ability to encode each view of a single object into a representation that indicates its view direction. We propose two models that employ different classification mechanisms; the first model selects an auto-associative network whose recovered view best matches the input view, and the second model is based on a modular architecture whose additional network classifies the views by splitting the input space nonlinearly. We demonstrate the effectiveness of the proposed classification models through simulations using 3D wire-frame objects.

## 1 INTRODUCTION

The human visual system can recognize various 3D (three-dimensional) objects from their 2D (two-dimensional) retinal images although the images vary significantly as the viewpoint changes. Recent computational models have explored how to learn to recognize 3D objects from their projected views (Poggio & Edelman, 1990). Most existing models are, however, based on supervised learning, i.e., during training the teacher tells which object each view belongs to. The model proposed by Weinshall et al. (1990) also requires a signal that segregates different objects during training. This paper, on the other hand, discusses unsupervised aspects of 3D object recognition where the system discovers categories by itself.

This paper presents an unsupervised classification scheme for categorizing 3D objects from their 2D views. The scheme consists of a mixture of 5-layer auto-associative networks, each of which identifies an object by non-linearly encoding the views into a representation that describes transformation of a rigid object. A mixture model with linear networks was also studied by Williams et al. (1993) for classifying objects under affine transformations. We propose two models that employ different classification mechanisms. The first model classifies the given view by selecting an auto-associative network whose recovered view best matches the input view. The second model is based on the modular architecture proposed by Jacobs et al. (1991) in which an additional 3-layer network classifies the views by directly splitting the input space. The simulations using 3D wire-frame objects demonstrate that both models effectively learn to classify each view as a 3D object.

This paper is organized as follows. Section 2 describes in detail the proposed models for unsupervised classification of 3D objects. Section 3 describes the simulation results using 3D wire-frame objects. In these simulations, we test the performance of the proposed models and examine what internal representations are acquired in the hidden layers. Finally, Section 4 concludes this paper.

## 2 THE NETWORK MODELS

This section describes an unsupervised scheme that classifies 2D views into 3D objects. We initially examined classical unsupervised clustering schemes, such as the k-means method or the vector quantization method, to see whether such methods can solve this problem (Duda & Hart, 1973). Through simulations using the wire-frame objects described in the next section, we found that these methods do not yield satisfactory performance. We, therefore, propose a new unsupervised learning scheme for classifying 3D objects.

The proposed scheme exploits an *auto-associative network* for identifying an object. An auto-associative network finds an identity mapping through a bottleneck in the hidden layer, i.e., the network approximates functions $F$ and $F^{-1}$ such that $\mathbf{R}^n \xrightarrow{F} \mathbf{R}^m \xrightarrow{F^{-1}} \mathbf{R}^n$ where m < n. The network, thus, compresses the input into a low dimensional representation by eliminating redundancy. If we use a five-layer perceptron network, the network can perform nonlinear dimensionality reduction, which is a nonlinear analogue to the principal component analysis (Oja, 1991; DeMers & Cottrell, 1993).

The proposed classification scheme consists of a mixture of five-layer auto-associative networks which we call *the identification networks*, or *the I-Nets*. In the case where the inputs are the projected views of a rigid object, the minimum dimension that constrains the input variation is *the degree of freedom* of the rigid object, which is six in the most general case, three for rotation and three for translation. Thus, a single I-Net can compress the views of an object into a representation whose dimension is its degree of freedom. The proposed scheme categorizes each view of a number of 3D objects into its class through selecting an appropriate I-Net. We present the following two models for different selection and learning methods.

**Model I:** The model I selects an I-Net whose output best fits the input (see Fig. 1). Specifically, we assume a classifier whose output vector is given by the softmax function of a negative squared difference between the input and the output of the I-Nets, i.e.,

$$f_i = \exp\left[-\|y*-y_i\|^2\right]\Big/\sum_j \exp\left[-\|y*-y_j\|^2\right] \tag{1}$$

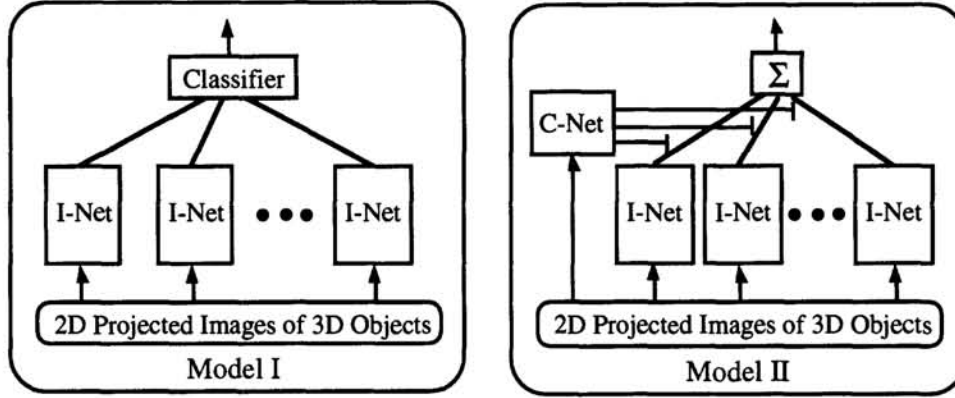

**Figure 1**: Model I and Model II. Each I-Net (identification net) is a 5-layer auto-associative network and the C-Net (classification net) is a 3-layer network.

where $y*$ and $y_i$ denote the input and the output of the $i$ th I-Net, respectively. Therefore, if only one of the I-Nets has an output that best matches the input, then the output value of the corresponding unit in the classifier becomes nearly one and the output values of the other units become nearly zero. For training the network, we maximize the following objective function:

$$\ln \frac{\sum_i \exp\left[-\alpha \|y*-y_i\|^2\right]}{\sum_i \exp\left[-\|y*-y_i\|^2\right]} \tag{2}$$

where $\alpha$ (>1) denotes a constant. This function forces the output of at least one I-Net to fit the input, and it also forces the rest of I-Nets to increase the error between the input and the output. Since it is difficult for a single I-Net to learn more than one object, we expect that the network will eventually converge to the state where each I-Net identifies only one object.

**Model II**: The model II, on the other hand, employs an additional network which we call *the classification network* or *the C-Net*, as illustrated in Fig. 1. The C-Net classifies the given views by directly partitioning the input space. This type of modular architecture has been proposed by Jacobs et al. (1991) based on a stochastic model (see also Jordan & Jacobs, 1992). In this architecture, the final output, $y$, is given by

$$y = \sum_i g_i y_i \tag{3}$$

where $y_i$ denotes the output of the $i$ th I-Net, and $g_i$ is given by the softmax function

$$g_i = \exp[s_i] \Big/ \sum_j \exp[s_j] \tag{4}$$

where $s_i$ is the weighted sum arriving at the $i$ th output unit of the C-Net.

For the C-Net, we use three-layer perceptron, since a simple perceptron with two layers did not provide a good performance for the objects used for our simulations (see Section

3). The results suggest that classification of such objects is not a linearly separable problem. Instead of using MLP (multi-layer perceptron), we could use other types of networks for the C-Net, such as RBF (radial basis function) (Poggio & Edelman, 1990).

We maximize the objective function

$$\ln \sum_i g_i \sigma^{-1} \exp\left[-\|y*-y_i\|^2 / (2\sigma^2)\right] \tag{5}$$

where $\sigma^2$ is the variance. This function forces the C-Net to select only one I-Net, and at the same time, the selected I-Net to encode and decode the input information.

Note that the model I can be interpreted as a modified version of the model II, since maximizing (2) is essentially equivalent to maximizing (5) if we replace $s_i$ of the C-Net in (4) with a negative squared difference between the input and the output of the $i$ th I-Net, i.e., $s_i = -\|y*-y_i\|^2$. Although the model I is a more direct classification method that exploits auto-associative networks, it is interesting to examine what information can be extracted from the input for classification in the model II (see Section 3.2).

## 3 SIMULATIONS

We implemented the network models described in the previous section to evaluate their performance. The 3D objects that we used for our simulations are 5-segment wire-frame objects whose six vertices are randomly selected in a unit cube, as shown in Fig. 2 (a) (see also Poggio & Edelman, 1990). Various views of the objects are obtained by orthographically projecting the objects onto an image plane whose position covers a sphere around the object (see Fig. 2 (b)). The view position is defined by the two parameters, $\theta$ and $\phi$. In the simulations, we used x, y image coordinates of the six vertices of three wire-frame objects for the inputs to the network.

The models contain three I-Nets, whose number is set equal to the number of the objects. The number of units in the third layer of the five-layer I-Nets is set equal to the number of the view parameters, which is two in our simulations. We used twenty units in the second and fourth layers. To train the network efficiently, we initially limited the ranges of $\theta$ and $\phi$ to $\pi/8$ and $\pi/4$ and gradually increased the range until it covered the whole sphere. During the training, objects were randomly selected among the three and their views were randomly selected within the view range. The steepest ascent method was used for maximizing the objective functions (2) and (5) in our simulations, but more efficient methods, such as the conjugate gradient method, can also be used.

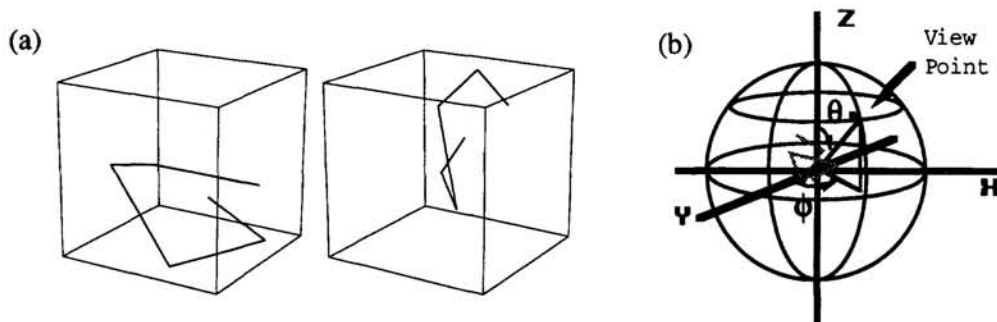

**Figure 2**: (a) 3D wire-frame objects. (b) Viewpoint defined by two parameters, $\theta$ and $\phi$.

## 3.1 SIMULATIONS USING THE MODEL I

This section describes the simulation results using the model I. As described in Section 2, the classifier of this model selects an I-Net that produces minimum error between the output and the input. We test the classification performance of the model and examine internal representations of the I-Nets after training the networks. The constant $\alpha$ in the objective function (2) was set to 50 during the training.

Fig. 3 shows the output of the classifier plotted over the view directions when the views of an object are used for the inputs. The output value of a unit is almost equal to one over the entire range of the view direction, and the outputs of the other two units are nearly zero. This indicates that the network effectively classifies a given view into an object regardless of the view directions. We obtained satisfactory results for classification if more than five units are used in the second and fourth layers of the I-Nets.

Fig. 4 shows examples of the input views of an object and the views recovered by the corresponding I-Net. The recovered views are significantly similar to the input views, indicating that each auto-associative I-Net can successfully compress and recover the views of an object. In fact, as shown in Fig. 5, the squared error between the input and the output of an I-Net is nearly zero for only one of the objects. This indicates that each I-Net can be used for identifying an object for almost entire view range.

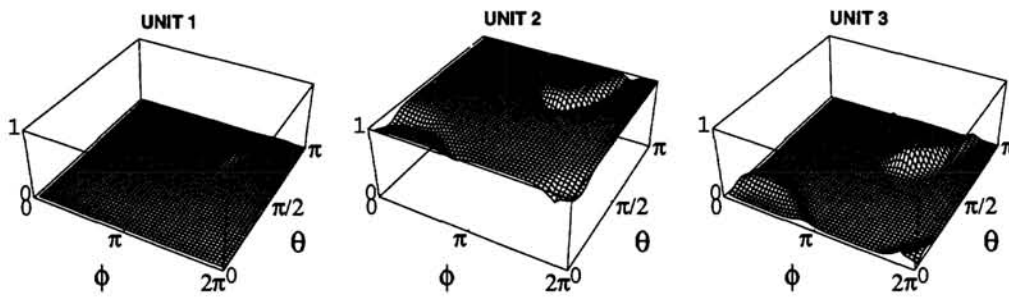

**Figure 3**: Outputs of the classifier in the model I. The output value of the second unit is almost equal to one over the full view range, and the outputs of the other two units are nearly zero for one of the 3D objects.

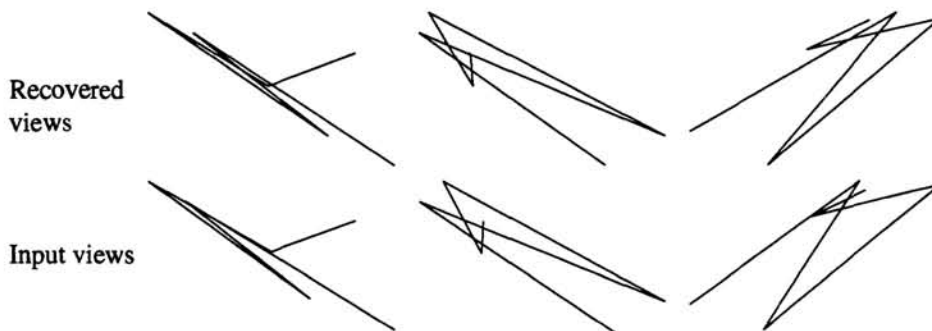

**Figure 4**: Examples of the input and recovered views of an object. The recovered views are significantly similar to the input views.

We further analyzed what information is encoded in the third layer of the I-Nets. Fig. 6 (a) illustrates the outputs of the third layer units plotted as a function of the view direction ($\theta$, $\phi$) of an object. Fig. 6 (b) shows the view direction ($\theta$, $\phi$) plotted as a function of the outputs of the third layer units. Both figures exhibit single-valued functions, i.e. the view direction of the object uniquely determines the outputs of the hidden units, and at the same time the outputs of the hidden units uniquely determine the view direction. Thus, each I-Net encodes a given view of an object into a representation that has one-to-one correspondence with the view direction. This result is expected from the condition that the dimension of the third layer is set equal to the degree of freedom of a rigid object.

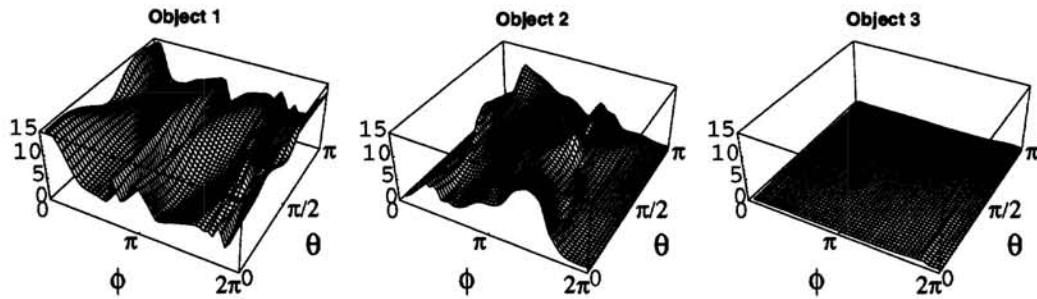

**Figure 5**: Error between the input view and the recovered view of an I-Net for each object. The figures show that the I-Net recovers only the views of Object 3.

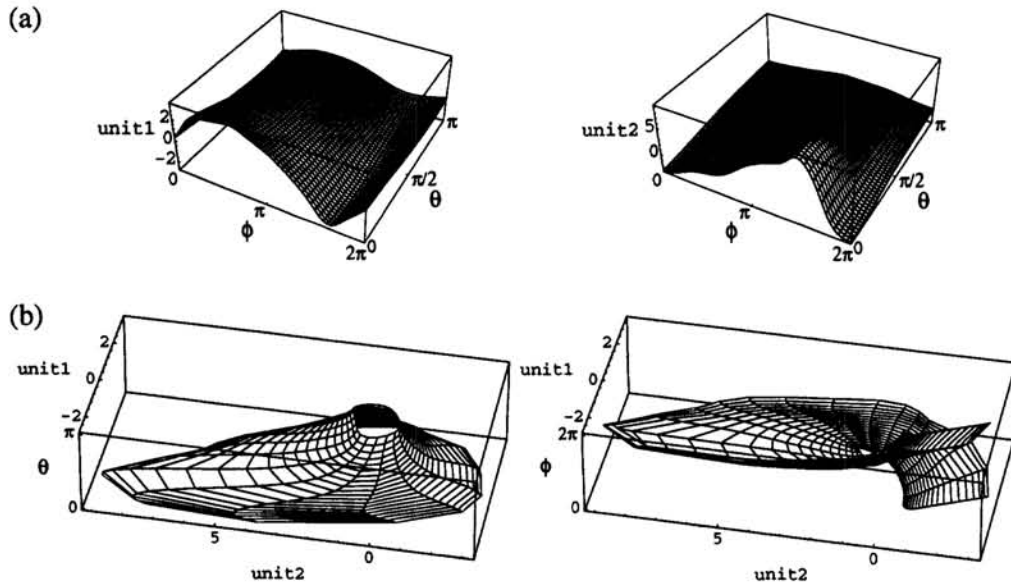

**Figure 6**: (a) Outputs of the third layer units of an I-Net plotted over the view direction ($\theta$, $\phi$) of an object. (b) The view direction plotted over the outputs of the third layer units. Figure (b) was obtained by inversely replotting Figure (a).

### 3.2 SIMULATIONS USING THE MODEL II

In this section, we show the simulation results using the model II. The C-Net in the model learns to classify the views by splitting the input space nonlinearly. We examine internal representations of the C-Net that lead to view invariant classification in its output.

In the simulations, we used the same 3 wire-frame objects used in the previous simulations. The C-Net contains 20 units in the hidden layer. The parameter $\sigma$ in the objective function (5) was set to 0.1. Fig. 7 (a) illustrates the values of an output unit in the C-Net for an object. As in the case of the model I, the model correctly classified the views into their original object for almost entire view range. Fig. 7 (b) illustrates the outputs of two of the hidden units as examples, showing that each hidden unit has a limited view range where its output is nearly one. The C-Net, thus, combines these partially invariant representations in the hidden layer to achieve full view invariance at the output layer.

To examine a generalization ability of the model, we limited the view range in the training period and tested the network using the images with the full view range. Fig. 8 (a) and (b) show the values of an output unit of the C-Net and the error of the corresponding I-Net plotted over the entire view range. The region surrounded by a rectangle indicates the range of view directions where the training was done. The figures show that the correct classification and the small recovery error are not restricted within the training range but spread across this range, suggesting that the network exhibits a satisfactory capability of generalization. We obtained similar generalization results for the model I as well. We also trained the networks with a sparse set of views rather than using randomly selected views. The results show that classification is nearly perfect regardless of the viewpoints if we use at least 16 training views evenly spaced within the full view range.

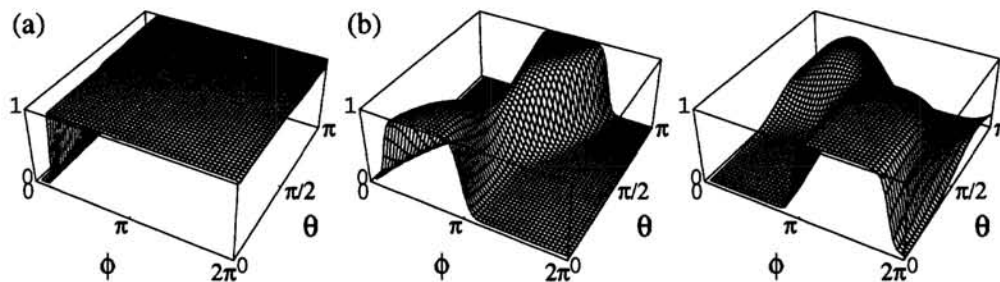

**Figure 7**: (a) Output values of an output unit of the C-Net when the views of an object are given (cf. Fig.3). (b) Output values of two hidden units of the C-Net for the same object.

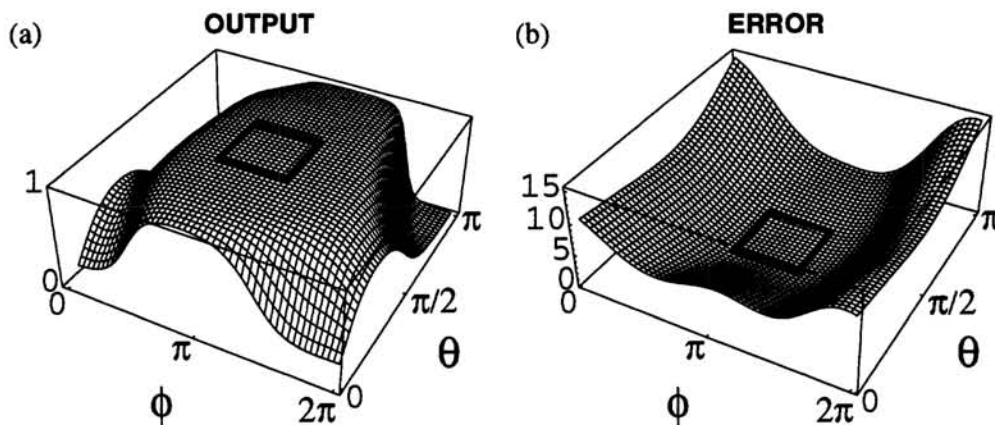

**Figure 8**: (a) Output values of an output unit of the C-Net. (b) Errors between the input views and the recovered views of the corresponding I-Net. The region surrounded by a rectangle indicates the view range where the training was done.

## 4  CONCLUSIONS

We have presented an unsupervised classification scheme that classifies 3D objects from their 2D views. The scheme consists of a mixture of non-linear auto-associative networks each of which identifies an object by encoding an input view into a representation that indicates its view direction. The simulations using 3D wire-frame objects demonstrated that the scheme effectively clusters the given views into their original objects with no explicit identification of the object classes being provided to the networks. We presented two models that utilize different classification mechanisms. In particular, the model I employs a novel classification and learning strategy that forces only one network to reconstruct the input view, whereas the model II is based on a conventional modular architecture which requires training of a separate classification network. Although we assumed in the simulations that feature points are already identified in each view and that their correspondence between the views is also established, the scheme does not, in principle, require the identification and correspondence of features, because the scheme is based solely on the existence of non-linear mappings between a set of images of an object and its degree of freedom. Therefore, we are currently investigating how the proposed scheme can be used to classify real gray-level images of 3D objects.

### Acknowledgments

We would like to thank Mitsuo Kawato for extensive discussions and continuous encouragement, and Hiroaki Gomi and Yasuharu Koike for helpful comments. We are also grateful to Tommy Poggio for insightful discussions.

## References

DeMers, D. and Cottrell, G. (1993). Non-linear dimensionality reduction. In Hanson, S. J., Cowan, J. D. & Giles, C. L., (eds), *Advances in Neural Information Processing Systems 5.* Morgan Kaufmann Publishers, San Mateo, CA. 580-587.

Duda, R. O. and Hart, P. E. (1973). *Pattern Classification and Scene Analysis.* John Wiley & Sons, NY.

Jacobs, R. A., Jordan, M. I., Nowlan, S. J. and Hinton, G. E. (1991). Adaptive mixtures of local experts. *Neural Computation,* 3, 79-87.

Jordan, M. I. and Jacobs, R. A. (1992). Hierarchies of adaptive experts. In Moody, J. E., Hanson, S. J. & Lippmann, R. P., (eds), *Advances in Neural Information Processing Systems 4.* Morgan Kaufmann Publishers, San Mateo, CA. 985-992.

Oja, E. (1991). Data compression, Feature extraction, and autoassociation in feedforward neural networks. In Kohonen, K. et al. (eds), *Artificial Neural Networks.* Elsevier Science publishers B.V., North-Holland.

Poggio, T. and Edelman, S. (1990). A network that learns to recognize three-dimensional objects. *Nature,* 343, 263.

Weinshall, D., Edelman, S. and Bülthoff, H. H. (1990). A self-organizing multiple-view representation of 3D objects. In Touretzky, D. S., (eds), *Advances in Neural Information Processing Systems 2.* Morgan Kaufmann Publishers, San Mateo, CA. 274-281.

Williams, C. K. I., Zemel, R. S. and Mozer, M. C. (1993). Unsupervised learning of object models. *AAAI Fall 1993 Symposium on Machine Learning in Computer Vision.*
